# Subject independent EEG-based BCI decoding

**Siamac Fazli**
Cristian Grozea
Márton Danóczy
Florin Popescu
Benjamin Blankertz
Klaus-Robert Müller

## Abstract

In the quest to make Brain Computer Interfacing (BCI) more usable, dry electrodes have emerged that get rid of the initial 30 minutes required for placing an electrode cap. Another time consuming step is the required individualized adaptation to the BCI user, which involves another 30 minutes calibration for assessing a subject's brain signature. In this paper we aim to also remove this calibration proceedure from BCI setup time by means of machine learning. In particular, we harvest a large database of EEG BCI motor imagination recordings (83 subjects) for constructing a library of subject-specific spatio-temporal filters and derive a subject independent BCI classifier. Our offline results indicate that BCI-naïve users could start real-time BCI use with no prior calibration at only a very moderate performance loss.

## 1 Introduction

The last years in BCI research have seen drastically reduced training and calibration times due to the use of machine learning and adaptive signal processing techniques (see [9] and references therein) and novel dry electrodes [18]. Initial BCI systems were based on operant conditioning and could easily require months of training on the subject side before it was possible to use them [1, 10]. Second generation BCI systems require to record a brief calibration session during which a subject assumes a fixed number of brain states, say, movement imagination and after which the subject-specific spatio-temporal filters (e.g. [6]) are inferred along with individualized classifiers [9]. Recently, first steps to transfer a BCI user's filters and classifiers between sessions was studied [14] and a further online-study confirmed that indeed such transfer is possible without significant performance loss [16]. In the present paper we even will go one step further in this spirit and propose a *subject-independent zero-training BCI* that enables both experienced and novice BCI subjects to use BCI immediately without calibration.

Our offline study applies a number of state of the art learning methods (e.g. SVM, Lasso etc.) in order to optimally construct such one-size-fits-all classifiers from a vast number of redundant features, here a large filter bank available from 83 BCI users. The use of sparsifying techniques specifically tell us what are the interesting aspects in EEG that are predictive to future BCI users. As expected, we find that a distribution of different alpha band features in combination with a number of characteristic common spatial patterns (CSPs) is highly predictive for all users. What is found as the outcome of a machine learning experiment can also be viewed as a compact quantitative description of the characteristic variability between individuals in the large subject group. Note that it is not the best subjects that characterize the variance necessary for a subject independent algorithm, rather the spread over existing physiology is to be represented concisely. Clearly, our proceedure may also be of use appart from BCI in other scientific fields, where complex characteristic features need to be homogenized into one overall inference model. The paper first provides an overview of the data used, then the ensemble learning algorithm is outlined, consisting of the procedure for building the

filters, the classifiers and the gating function, where we apply various machine learning methods. Interestingly we are able to successfully classify trials of novel subjects with zero training suffering only a small loss in performance. Finally we put our results into perspective.

## 2 Available Data and Experiments

We used 83 BCI datasets (sessions), each consisting of 150 trials from 83 individual subjects. Each trial consists of one of two predefined movement imaginations, being left and right hand, i.e. data was chosen such that it relies only on these 2 classes, although originally three classes were cued during the calibration session, being left hand (L), right hand (R) and foot (F). 45 EEG channels, which are in accordance with the 10-20 system, were identified to be common in all sessions considered. The data were recorded while subjects were immobile, seated on a comfortable chair with arm rests. The cues for performing a movement imagination were given by visual stimuli, and occurred every 4.5-6 seconds in random order. Each trial was referenced by a 3 second long time-window starting at 500 msec after the presentation of the cue. Individual experiments consisted of three different training paradigms. The first two training paradigms consisted of visual cues in form of a letter or an arrow, respectively. In the third training paradigm the subject was instructed to follow a moving target on the screen. Within this target the edges lit up to indicate the type of movement imagination required. The experimental proceedure was designed to closely follow [3]. Electromyogram (EMG) on both forearms and the foot were recorded as well as electrooculogram (EOG) to ensure there were no real movements of the arms and that the movements of the eyes were not correlated to the required mental tasks.

## 3 Generation of the Ensemble

The ensemble consists of a large redundant set of subject-dependent common spatial pattern filters (CSP cf. [6]) and their matching classifiers (LDA). Each dataset is first preprocessed by 18 predefined temporal filters (i.e. band-pass filters) in parallel (see upper panel of Figure 1). A corresponding spatial filter and linear classifier is obtained for every dataset and temporal filter. Each resulting CSP-LDA couple can be interpreted as a potential basis function. Finding an appropriate weighting for the classifier outputs of these basis functions is of paramount importance for the accurate prediction. We employed different forms of regression and classification in order to find an optimal weighting for predicting the movement imagination data of unseen subjects[2, 4]. This processing was done by leave-one-subject-out cross-validation, i.e. the session of a particular subject was removed, the algorithm trained on the remaining trials (of the other subjects) and then applied to this subject's data (see lower panel of Figure 1).

### 3.1 Temporal Filters

The $\mu$-rhythm (9-14 Hz) and synchronized components in the $\beta$-band (16-22 Hz) are macroscopic idle rhythms that prevail over the postcentral somatosensory cortex and precentral motor cortex, when a given subject is at rest. Imaginations of movements as well as actual movements are known to suppress these idle rhythms contralaterally. However, there are not only subject-specific differences of the most discriminative frequency range of the mentioned idle-rhythms, but also session differences thereof.
We identified 18 neurophysiologically relevant temporal filters, of which 12 lie within the $\mu$-band, 3 in the $\beta$-band, two in between $\mu$- and $\beta$-band and one broadband $7 - 30$Hz. In all following performance related tables we used the percentage of misclassified trials, or 0-1 loss.

### 3.2 Spatial Filters and Classifiers

CSP is a popular algorithm for calculating spatial filters, used for detecting event-related (de-)synchronization (ERD/ERS), and is considered to be the gold-standard of ERD-based BCI systems [13, 19, 6]. The CSP algorithm maximizes the variance of right hand trials, while simultaneously minimizing the variance for left hand trials. Given the two covariance matrices $\Sigma_1$ and $\Sigma_2$, of size *channels* x *concatenated timepoints*, the CSP algorithm returns the matrices $W$ and $D$. $W$ is a matrix of projections, where the i-th row has a relative variance of $d_i$ for trials of class 1 and a relative

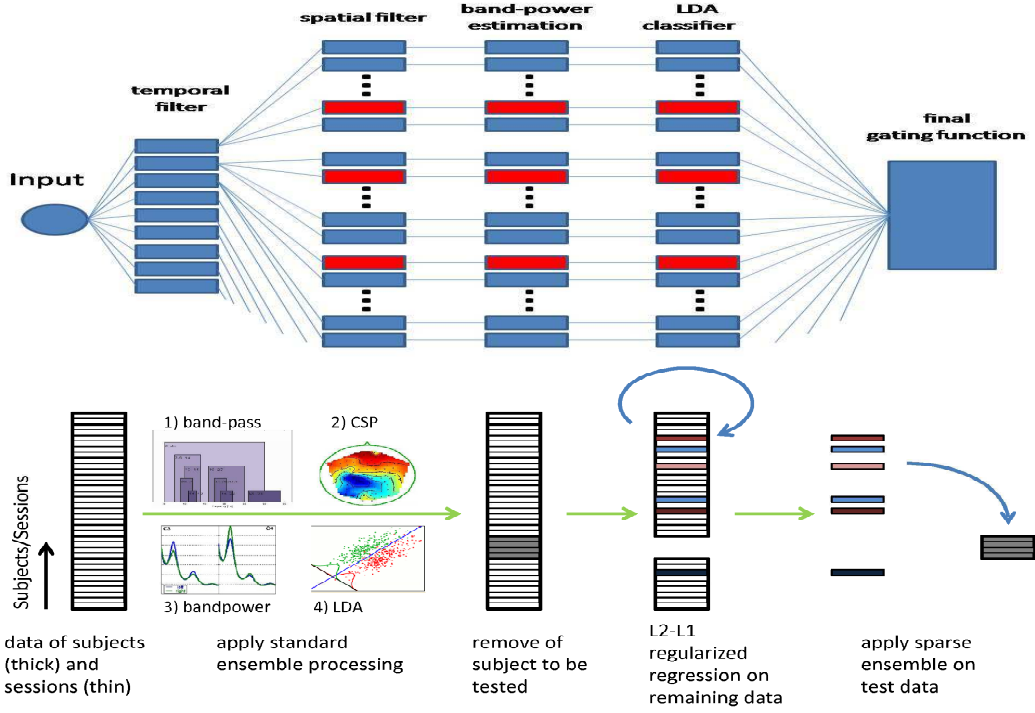

Figure 1: 2 Flowcharts of the ensemble method. The red patches in the top panel illustrate the inactive nodes of the ensemble after sparsification.

variance of $1 - d_i$ for trials of class 2. $D$ is a diagonal matrix with entries $d_i \in [0, 1]$, with length $n$, the number of channels:

$$W\Sigma_1 W^T = D \qquad \text{and} \qquad W\Sigma_2 W^T = I - D \qquad (1)$$

Best discrimination is provided by filters with very high (emphazising one class) or very low eigenvalues (emphazising the other class), we therefore chose to only include projections with the highest 2 and corresponding lowest 2 eigenvalues for our analysis. We use Linear Discriminant Analysis (LDA) [5], each time filtered session corresponds to a CSP set and to a matched LDA.

### 3.3 Final gating function

The final gating function combines the outputs of the individual ensemble members to a single one. This can be realized in many ways. For a number of ensemble methods the mean has proven to be a surprisingly good choice [17]. As a baseline for our ensemble we simply averaged all outputs of our individual classifiers. This result is given as *mean* in Table 2.

**Classification** We employ various classification methods such as k Nearest Neighbor (kNN), Linear Discriminant Analysis (LDA), Support Vector Machine (SVM) and a Linear Programming Machine (LPM) [12].

**Quadratic regression with $\ell_1$ regularization**

$$\operatorname*{argmin}_{w_{ij}^{(k)}} \sum_{x \in X \setminus X_k} (h_k(x) - y(x))^2 + \alpha \sqrt{\sum_{i=1}^{B} \sum_{j \in S \setminus S_k} \sum_{x \in X \setminus X_k} c_{ij}(x)^2} \left( \sum_{i=1}^{B} \sum_{j \in S \setminus S_k} |w_{ij}^{(k)}| + |b| \right) \qquad (2)$$

$$h_k(x) = \sum_{i=1}^{B} \sum_{j \in S \setminus S_k} w_{ij}^{(k)} c_{ij}(x) - b \qquad (3)$$

where $c_{ij}(x) \in [-\infty; \infty]$ is the continuous classifier output, before thresholding, obtained from the session $j$ by applying the bandpass filter $i$, $B$ is the number of frequency bands, $S$ the complete set

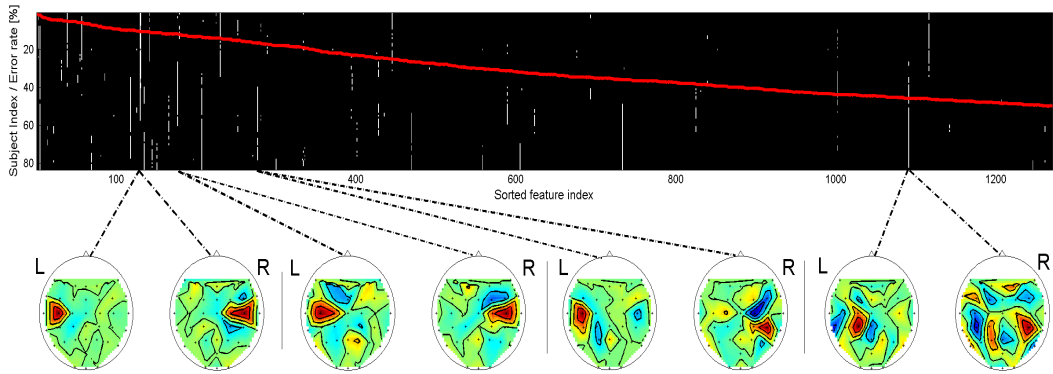

Figure 2: Feature selection during cross-validation: white dashes mark the features kept after regularization for the prediction of the data of each subject. The numbers on the vertical axis represent the subject index as well as the Error Rate (%). The red line depicts the baseline error of individual subjects (classical auto-band CSP). Features as well as baseline errors are sorted by the error magnitude of the self-prediction. Note that some of the features are useful in predicting the data of most other subjects, while some are rarely or never used.

of sessions, $X$ the complete data set, $S_k$ the set of sessions of subject $k$, $X_k$ the dataset for subject $k$, $y(x)$ is the class label of trial $x$ and $w_{ij}^k$ in equation (3) are the weights given to the LDA outputs. The hyperparameter $\alpha$ in equation (2) was varied on a logarithmic scale and multiplied by a dataset scaling factor which accounted for fluctuations in voting population distribution and size for each subject. The dataset scaling factor is computed using $c_{ij}(x)$, for all $x \in X \setminus X_k$. For computational efficiency reasons the hyperparameter was tuned on a small random subset of subjects whose labels are to be predicted from data obtained from other subjects such that the resulting test/train error ratio was minimal, which in turn affected the choice (leave in/out) of classifiers among the 83x18 candidates. The $\ell_1$ regularized regression with this choice of $\alpha$ was then applied to all subjects, with results (in terms of feature sparsification) shown in Figure 2. In fact the exemplary CSP patterns shown in the lower part of the Figure exhibit neurophysiologically meaningful activation in motor-cortical areas. The most predictive subjects show smooth monopolar patterns, while subjects with a higher self-prediction loss slowly move from dipolar to rather ragged maps. From the point of view of approximation even the latter make sense for capturing the overall ensemble variance.

The implementation of the regressions were performed using CVX, a package for specifying and solving convex programs [11]. We coupled an $\ell_2$ loss with an $\ell_1$ penalty term on a linear voting scheme ensemble.

**Least Squares Regression** Is a special case of equation (2), with $\alpha = 0$.

## 3.4 Validation

The subject-specific CSP-based classification methods with automatically, subject-dependent tuned temporal filters (termed reference methods) are validated by an 8-fold cross-validation, splitting the data chronologically. The chronological splitting for cross-validation is a common practice in EEG classification, since the non-stationarity of the data is thus preserved [9].

To validate the quality of the ensemble learning we employed a leave-one-subject out cross-validation (LOSO-CV) procedure, i.e. for predicting the labels of a particular subject we only use data from other subjects.

## 4 Results

Overall performance of the reference methods, other baseline methods and of the ensemble method is presented in Table 2. Reference method performances of subject-specific CSP-based classification are presented with heuristically tuned frequency bands [6]. Furthermore we considered much simpler (zero-training) methods as a control. Laplacian stands for the power difference in two Laplace filtered channels (C3 vs. C4) and simple band-power stands for the power difference of the same two

| | classification | | | | regression | |
|---|---|---|---|---|---|---|
| % of data | kNN | LDA | LPM | SVM | LSR | LSR-$\ell_1$ |
| 10 | 31.3 | 45.3 | 37.3 | 31.3 | 46.0 | 30.7 |
| 20 | 32.0 | 40.0 | 38.0 | **28.7** | 42.0 | 31.3 |
| 30 | 32.7 | 38.7 | 37.3 | 33.1 | 38.0 | 30.0 |
| 40 | 32.7 | 36.0 | 37.9 | 31.3 | 36.7 | **29.3** |

Table 1: Main results of various machine learning algorithms.

| approach | machine learning | | | | | | | classical | | |
|---|---|---|---|---|---|---|---|---|---|---|
| | zero training | | | | | | | | | training |
| method | mean | kNN | LDA | LPM | SVM | LSR | LSR-$\ell_1$ | Lap | BP | CSP |
| # $<25\%$ | 31 | 30 | 18 | 14 | 29 | 19 | **36** | 24 | 11 | **39** |
| 25%-tile | 17.3 | 17.3 | 27.3 | 26.7 | 18.7 | 26.0 | 16.0 | 22.0 | 31.3 | 11.9 |
| median | 30.7 | 31.3 | 36.0 | 37.3 | **28.7** | 36.7 | **29.3** | 34.7 | 38.7 | **25.9** |
| 75%-tile | 41.3 | 42.0 | 43.3 | 44.0 | 41.3 | 44.0 | 40.7 | 45.3 | 45.3 | 41.4 |

Table 2: Comparing ML results to various baselines.

channels without any spatial filtering. For the simple zero-training methods we chose a broad-band filter of $7-30$Hz, since it is the least restrictive and scored one of the best performances on a subject level. The bias $b$ in equation (3) can be tuned broadly for all sessions or corrected individually by session, and implemented for online experiments in multiple ways [16, 20, 15]. In our case we chose to adapt $b$ without label information, but operating under the assumption that class frequency is balanced. We therefore simply subtracted the mean over all trials of a given session. Table 1 shows a comparison of the various classification schemes. We evaluate the performance on a given percentage of the training data in order to observe information gain as a function of datapoints. Clearly the two best ML techniques are on par with subject-dependent CSP classifiers and outperform the simple zero-training methods (not shown in Table 1 but in Table 2) by far. While SVM scores the best median loss over all subjects (see Table 1), L1 regularized regression scored better results for well performing BCI subjects (Figure 3 column 1, row 3). In Figure 3 and Table 2 we furthermore show the results of the L1 regularized regression and SVM versus the auto-band reference method (zero-training versus subject-dependent training) as well as vs. the simple zero-training methods Laplace and band-power. Figure 4 shows all individual temporal filters used to generate the ensemble, where their color codes for the frequency they were used to predict labels of previously unseen data. As to be expected mostly $\mu$-band related temporal filters were selected. Contrary to what one may expect, features that generalize well to other subjects' data do not exclusively come from BCI subjects with low self-prediction errors (see white dashes in Figure 2), in fact there are *some* features of weak performing subjects that are necessary to capture all variance of the ensemble. However there is a strong correlation between subjects with a low self-prediction loss and the generalizability of their features to predicting other subjects, as can be seen on the right part of Figure 4.

## 4.1 Focusing on a particular subject

In order to give an intuition of how the ensemble works in detail we will focus on a particular subject. We chose to use the subject with the lowest reference method cross-validation error (10%). Given the non-linearity in the band-power estimation (see Figure 1) it is impossible to picture the resulting ensemble spatial filter exactly. However, by averaging the chosen CSP filters with the weightings, obtained by the ensemble and multiplying them by their LDA classifier weight, we get an approximation:

$$P_{ENS} = \sum_{i=1}^{B} \sum_{j \in S \setminus S_k} w_{ij} W_{ij} C_{ij} \tag{4}$$

where $w_{ij}$ is the weight matrix, resulting from the $\ell_1$ regularized regression, given in equations (2) and (3), $W_{ij}$ the CSP filter, corresponding to temporal filter i and subject $j$ and $C_{ij}$ the LDA weights (B in Figure 5). For the case of classical auto-band CSP this simply reduces to $P_{CSP} = WC$ (A in Figure 5). Another way to exemplify the ensemble performance is to refer to a transfer function. By injecting a sinusoid with a frequency within the corresponding band-pass filter into a given channel

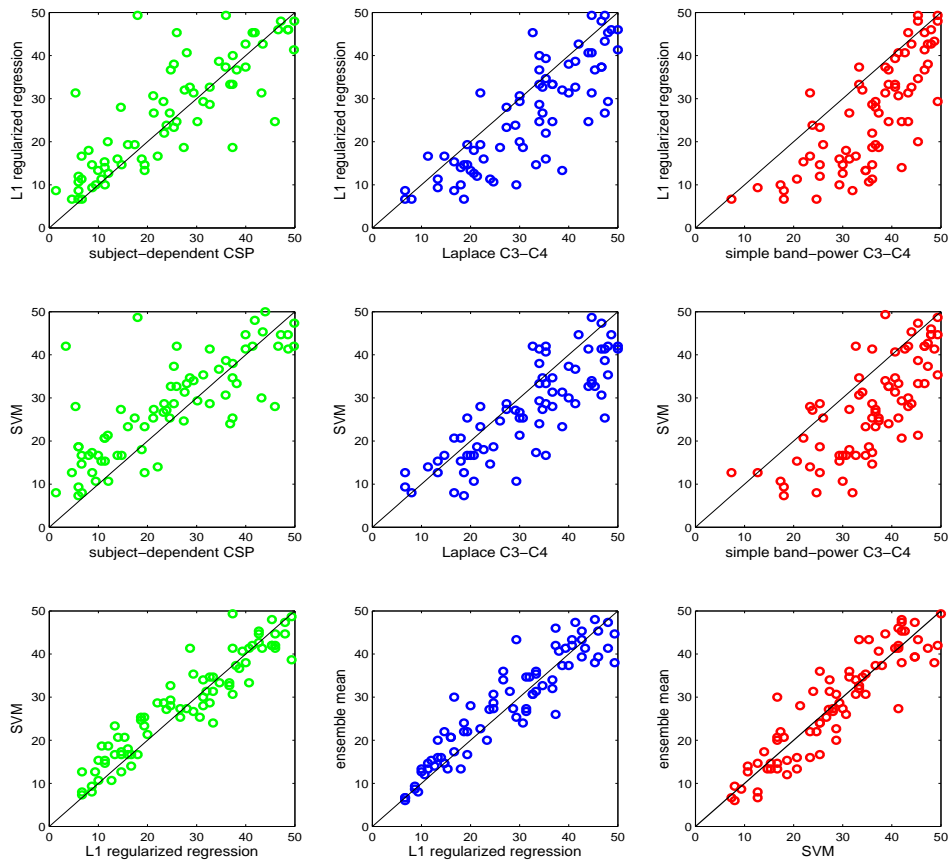

Figure 3: Compares the two best-scoring machine learning methods $\ell_1$-regularized regression and Support Vector Machine to subject-dependent CSP and other simple zero-training approaches. The axis show the classification loss in percent.

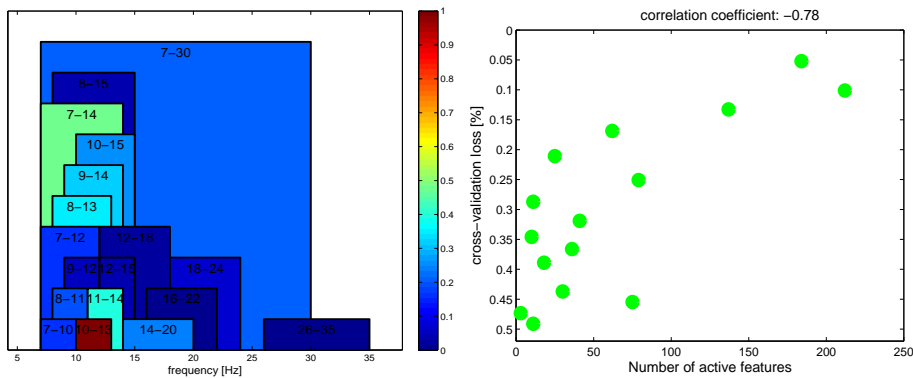

Figure 4: On the left: The used temporal filters and in color-code their contribution to the final L1 regularized regression classification (the scale is normalized from 0 to 1). Cleary $\mu$-band temporal filters between $10 - 13$Hz are most predictive. On the right: Number of features used vs. self-predicted cross-validation. A high self-prediction can be seen to yield a large number of features that are predictable for the whole ensemble.

and processing it by the four CSP filters, estimating the bandpower of the resulting signal and finally combining the four outputs by the LDA classifier, we obtain a response for the particular channel, where the sinusoid was injected. Repeating this procedure for each channel results in a response matrix. This procedure can be applied for a single CSP/LDA pair, however we may also repeat the given method for as many times as features were chosen for a given subject by the ensemble and hence obtain an accurate description of how the ensemble processes the given EEG data. The resulting response matrices are displayed in panel C of Figure 5. While the subject-specific pattern (classical) looks less focused and more diverse the general pattern matches the one obtained by the ensemble. A third way of visualizing how the ensemble works, we show the primary projections of the CSP filters that were given the 6 highest weights by the ensemble on the left panel (F) and the distribution of all weights in panel D. The spatial positions of highest channel weightings differ slightly for each of the CSP filters given, however the maxima of the projection matrices are clearly positioned around the primary motor cortex.

## 5   Conclusion

On the path of bringing BCI technology from the lab to a more practical every day situation, it becomes indispensable to reduce the setup time which is nowadays more than one hour towards less than one minute. While dry electrodes provide a first step to avoid the time for placing the cap, calibration still remained and it is here where we contribute by dispensing with calibration sessions. Our present study is an offline analysis providing a positive answer to the question whether a *subject independent* classifier could become reality for a BCI-naïve user. We have taken great care in this work to exclude data from a given subject when predicting his/her performance by using the previously described LOSOCV. In contrast with previous work on ensemble approaches to BCI classification based on simple majority voting and Adaboost [21, 8] that have utilized only a limited dataset, we have profited greatly by a large body of high quality experimental data accumulated over the years. This has enabled us to choose by means of machine learning technology a very sparse set of voting classifiers which performed as well as standard, state-of-the-art subject calibrated methods. $\ell_1$ regularized regression in this case performed better than other methods (such as majority voting) which we have also tested. Note that, interestingly, the chosen features (see Figure 2), do not exclusively come from the best performing subjects, in fact some average performer was also selected. However most white dashes are present in the left half, i.e. most subjects with high auto-band reference method performance were selected. Interestingly some subjects with very high BCI performance are not selected at all, while others generalize well in the sense that their model are able to predict other subject's data. No single frequency band dominated classification accuracy – see Figure 4. Therefore, the regularization must have selected diverse features. Nevertheless, as can be seen in panel G of Figure 5 there is significant redundancy between classifiers in the ensemble. Our approach of finding a sparse solution reduces the dimensionality of the chosen features significantly. For very able subjects our zero-training method exhibits a slight performance decrease, which however will not prevent them from performing successfully in BCI. The sparsification of classifiers, in this case, also leads to potential insight into neurophysiological processes. It identifies relevant cortical locations and frequency bands of neuronal population activity which are in agreement with general neuroscientific knowledge. While this work concentrated on zero training classification and not brain activity interpretation, a much closer look is warranted. Movement imagination detection is not only determined by the cortical representation of the limb whose control is being imagined (in this case the arm) but also by differentially located cortical regions involved in movement planning (frontal), execution (fronto-parietal) and sensory feedback (occipito-parietal). Patterns relevant to BCI detection appear in all these areas and while dominant discriminant frequencies are in the $\alpha$ range, higher frequencies appear in our ensemble, albeit in combination with less focused patterns. So what we have found from our machine learning algorithm should be interpreted as representing the characteristic neurophysiological variation a large subject group, which in itself is a highly relevant topic that goes beyond the scope of this technical study. Future online studies will be needed to add further experimental evidence in support of our findings. We plan to adopt the ensemble approach in combination with a recently developed EEG cap having dry electrodes [18] and thus to be able to reduce the required preparation time for setting up a running BCI system to essentially zero. The generic ensemble classifier derived here is also an excellent starting point for a subsequent coadaptive learning procedure in the spirit of [7].

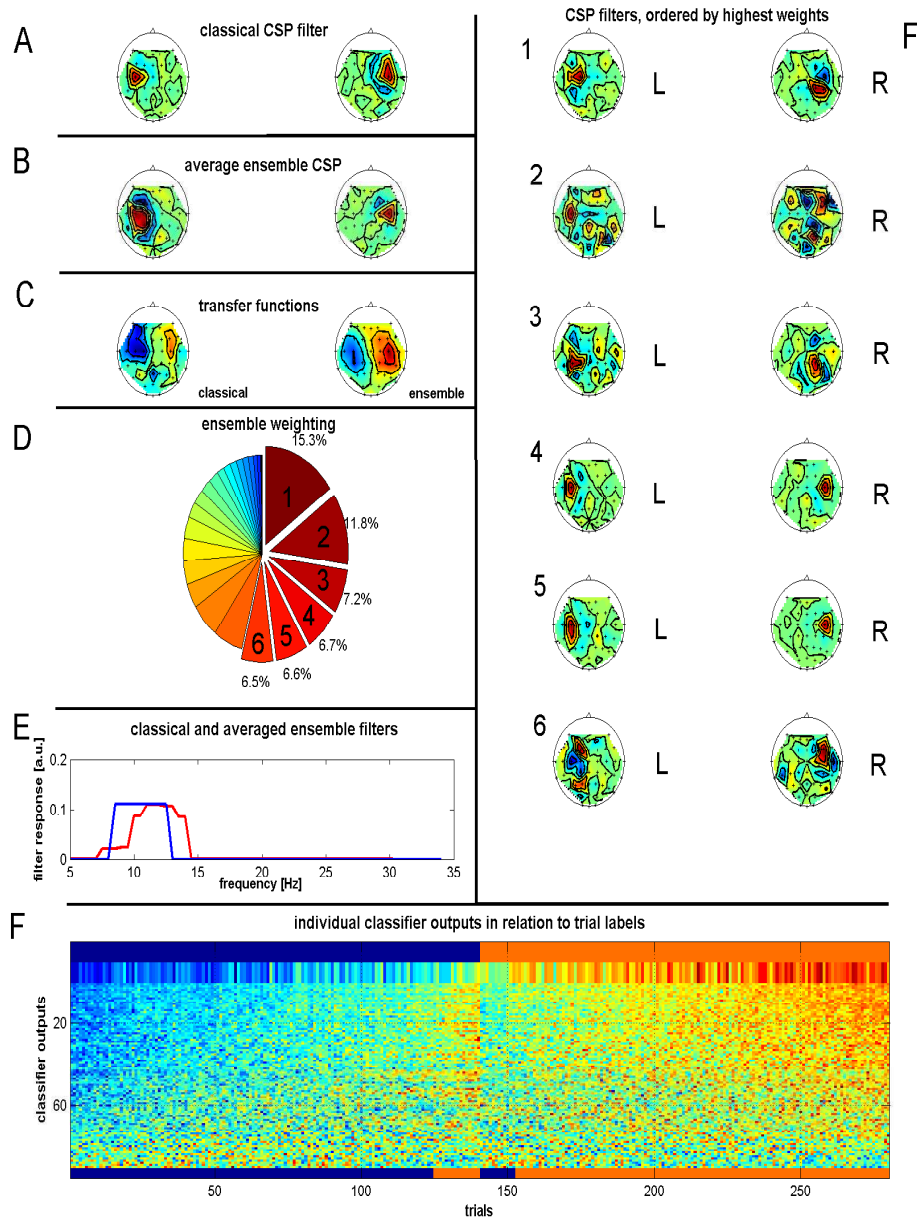

Figure 5: A: primary projections for classical auto-band CSP. B: linearly averaged CSP's from the ensemble. C: transfer function for classical auto-band and ensemble CSP's. D: weightings of 28 ensemble members, the six highest components are shown in F. E: linear average ensemble temporal filter (red), heuristic (blue). F: primary projections of the 6 ensemble members that received highest weights. G: Broad-band version of the ensemble for a single subject. The outputs of all basis classifiers are applied to each trial of one subject. The top row (broad) gives the label, the second row (broad) gives the output of the classical auto-band CSP, and each of the following rows (thin) gives the outputs of the individual classifiers of other subjects. The individual classifier outputs are sorted by their correlation coefficient with respect to the class labels. The trials (columns) are sorted by true labels with primary key and by mean ensemble output as a secondary key. The row at the bottom gives the sign of the average ensemble output.

# References

[1] N. Birbaumer, N. Ghanayim, T. Hinterberger, I. Iversen, B. Kotchoubey, A. Kübler, J. Perelmouter, E. Taub, and H. Flor. A spelling device for the paralysed. *Nature*, 398:297–298, 1999.

[2] B. Blankertz, G. Curio, and K.-R. Müller. Classifying single trial EEG: Towards brain computer interfacing. In T. G. Diettrich, S. Becker, and Z. Ghahramani, editors, *Advances in Neural Inf. Proc. Systems (NIPS 01)*, volume 14, pages 157–164, 2002.

[3] B. Blankertz, G. Dornhege, M. Krauledat, K.-R. Müller, V. Kunzmann, F. Losch, and G. Curio. The Berlin Brain-Computer Interface: EEG-based communication without subject training. *IEEE Trans Neural Syst Rehabil Eng*, 14:147–152, 2006.

[4] B. Blankertz, G. Dornhege, S. Lemm, M. Krauledat, G. Curio, and K.-R. Müller. The Berlin Brain-Computer Interface: Machine learning based detection of user specific brain states. *Journal of Universal Computer Science*, 12:2006, 2006.

[5] B. Blankertz, G. Dornhege, C. Schäfer, R. Krepki, J. Kohlmorgen, K.-R. Müller, V. Kunzmann, F. Losch, and G. Curio. Boosting bit rates and error detection for the classification of fast-paced motor commands based on single-trial EEG analysis. *IEEE Trans. Neural Sys. Rehab. Eng.*, 11(2):127–131, 2003.

[6] B. Blankertz, R. Tomioka, S. Lemm, M. Kawanabe, and K.-R. Müller. Optimizing spatial filters for robust EEG single-trial analysis. *IEEE Signal Proc Magazine*, 25(1):41–56, 2008.

[7] Benjamin Blankertz and Carmen Vidaurre. Towards a cure for bci illiteracy: Machine-learning based co-adaptive learning. *BMC Neuroscience*, 10, 2009.

[8] R. Boostani and M. H. Moradi. A new approach in the BCI research based on fractal dimension as feature and adaboost as classifier. *J. Neural Eng.*, 1:212–217, 2004.

[9] G. Dornhege, J.del R. Millán, T. Hinterberger, D. McFarland, and K.-R. Müller, editors. *Toward Brain-Computer Interfacing*. Cambridge, MA: MIT Press, 2007.

[10] T. Elbert, B. Rockstroh, W. Lutzenberger, and N. Birbaumer. Biofeedback of slow cortial potentials. I. *Electroencephalogr. Clin. Neurophysiol.*, 48:293–301, 1980.

[11] M. Grant and S. Boyd. CVX: Matlab software for disciplined convex programming (web page and software). *http://stanford.edu/ boyd/cvx*, 2008.

[12] Trevor Hastie, Robert Tibshirani, and Jerome Friedman. *The Elements of Statistical Learning, Second Edition: Data Mining, Inference, and Prediction (Springer Series in Statistics)*. Springer New York, 2 edition, 2001.

[13] Z.J. Koles and A. C. K. Soong. EEG source localization: implementing the spatio-temporal decomposition approach. *Electroencephalogr. Clin Neurophysiol*, 107:343–352, 1998.

[14] M. Krauledat, M. Schröder, B. Blankertz, and K.-R. Müller. Reducing calibration time for brain-computer interfaces: A clustering approach. In B. Schölkopf, J. Platt, and T. Hoffman, editors, *Advances in Neural Inf. Proc. Systems (NIPS 07)*, volume 19, pages 753–760, 2007.

[15] M. Krauledat, P. Shenoy, B. Blankertz, R.P.N. Rao, and K.-R. Müller. Adaptation in csp-based BCI systems. In *Toward Brain-Computer Interfacing*, pages 305–309. MIT Press, 2007.

[16] M. Krauledat, M. Tangermann, B. Blankertz, and K.-R. Müller. Towards zero training for brain-computer interfacing. *PLoS ONE*, 3:e2967, 2008.

[17] R. Polikar. Ensemble based systems in decision making. *IEEE Circuits and Systems Magazine*, 6(3):21–45, 2006.

[18] F. Popescu, S. Fazli, Y. Badower, B. Blankertz, and K.-R. Müller. Single trial classification of motor imagination using 6 dry EEG electrodes. *PLoS ONE*, 2:e637, 2007.

[19] H. Ramoser, J. Müller-Gerkin, and G. Pfurtscheller. Optimal spatial filtering of single trial EEG during imagined hand movement. *IEEE Trans. Rehab. Eng*, 8(4):441–446, 2000.

[20] P. Shenoy, M. Krauledat, B. Blankertz, R.P.N. Rao, and K.-R. Müller. Towards adaptive classification for BCI. *Journal of Neural Engineering*, 3(1):R13–R23, 2006.

[21] S. Wang, Z. Lin, and C. Zhang. Network boosting for BCI applications. *Book Series Lecture Notes in Computer Science*, 3735:386–388, 2005.

